# Bayesian Exponential Family PCA

**Shakir Mohamed**      **Katherine Heller**      **Zoubin Ghahramani**
Department of Engineering, University of Cambridge
Cambridge, CB2 1PZ, UK
{sm694,kah60,zoubin}@eng.cam.ac.uk

## Abstract

Principal Components Analysis (PCA) has become established as one of the key tools for dimensionality reduction when dealing with real valued data. Approaches such as exponential family PCA and non-negative matrix factorisation have successfully extended PCA to non-Gaussian data types, but these techniques fail to take advantage of Bayesian inference and can suffer from problems of overfitting and poor generalisation. This paper presents a fully probabilistic approach to PCA, which is generalised to the exponential family, based on Hybrid Monte Carlo sampling. We describe the model which is based on a factorisation of the observed data matrix, and show performance of the model on both synthetic and real data.

## 1   Introduction

In Principal Components Analysis (PCA) we seek to reduce the dimensionality of a $D$-dimensional data vector to a smaller $K$-dimensional vector, which represents an embedding of the data in a lower dimensional space. The traditional PCA algorithm is non-probabilistic and defines the eigenvectors corresponding to the $K$-largest eigenvalues as this low dimensional embedding. In probabilistic approaches to PCA, such as probabilistic PCA (PPCA) and Bayesian PCA [1], the data is modelled by unobserved latent variables, and these latent variables define the low dimensional embedding. In these models both the data and the latent variables are assumed to be Gaussian distributed.

This Gaussian assumption may not be suitable for all data types, especially in the case where data is binary or integer valued. Models such as Non-negative Matrix Factorisation (NMF) [2], Discrete Components Analysis (DCA) [3], Exponential Family PCA (EPCA) [4] and Semi-parametric PCA (SP-PCA) [5], have been developed that endow PCA the ability to handle data for which Bernoulli or Poisson distributions may be more appropriate. These general approaches to PCA involve the representation of the data matrix $\mathbf{X}$ as a product of smaller matrices: the factor score matrix $\mathbf{V}$, representing the reduced vectors; and a data independent part $\mathbf{\Theta}$, known as the factor loading matrix. In the original data matrix, there are $N \times D$ entries, and in the matrix factorisation there are $(N + D) \times K$ entries, which is a reduction in the number of parameters if $K \ll N, D$ [3].

Models such as PCA, NMF and EPCA are from the class of deterministic latent variable models [6], since their latent variables are set to their maximum a posteriori (MAP) values. Welling et al. [6] argue that the resulting model essentially assigns zero probability to all input configurations that are not in the training set. This problem stems from the use of an inappropriate objective function, and can be remedied by using an alternate approximate inference scheme. In this paper, we propose a fully Bayesian approach to PCA generalised to the exponential family.

Our approach follows the method of factorising the data matrix into two lower rank matrices using an exponential family distribution for the data with conjugate priors. The exponential family of distributions is reviewed in section 2, and the complete specification for the model is given in section 3. Learning and inference in the model is performed using the Hybrid Monte Carlo approach, which is

appropriate due to the continuous nature of variables in the model. The connections to existing generalised PCA methods, such as NMF and EPCA are discussed in section 4. We present results on the performance of our Bayesian exponential family PCA model in section 5. We report performance using both a synthetic data set to highlight particular model properties and also on two real datasets: the Cedar Buffalo digits dataset and data on cardiac SPECT images. The Bayesian approach gives us many samples of the final low dimensional embedding of the data, and techniques for determining a single low dimensional embedding are discussed in section 6. In section 7 we conclude, and present a survey of possible future work.

## 2  Exponential Family Models

In the exponential family of distributions, the conditional probability of a value $\mathbf{x}_n$ given parameter value $\boldsymbol{\theta}$, takes the following form:

$$p(\mathbf{x}_n|\boldsymbol{\theta}) = \exp\{s(\mathbf{x}_n)^\top\boldsymbol{\theta} + h(\mathbf{x}_n) + g(\boldsymbol{\theta})\} \tag{1}$$

where $s(\mathbf{x}_n)$ are the sufficient statistics, $\boldsymbol{\theta}$ is a vector of natural parameters, $h(\mathbf{x}_n)$ is a function of the data and $g(\boldsymbol{\theta})$ is a function of the parameters. In this paper, the natural representation of the exponential family likelihood is used, such that $s(\mathbf{x}_n) = \mathbf{x}_n$.

It is convenient to represent a variable $\mathbf{x}_n$ that is drawn from an exponential family distribution using the notation: $\mathbf{x}_n \sim \text{Expon}(\boldsymbol{\theta})$ with natural parameters $\boldsymbol{\theta}$. Probability distributions that belong to the exponential family also have natural conjugate prior distributions $p(\boldsymbol{\theta})$. The conjugate prior distribution for the exponential family distribution of equation (1) is:

$$p(\boldsymbol{\theta}) \propto \exp\{\boldsymbol{\lambda}^\top\boldsymbol{\theta} + \nu g(\boldsymbol{\theta}) + f(\boldsymbol{\lambda})\} \tag{2}$$

where $\boldsymbol{\lambda}$ and $\nu$ are hyperparameters of the prior distribution. In this case we use the notation: $\boldsymbol{\theta} \sim \text{Conj}(\boldsymbol{\lambda}, \nu)$ as shorthand for the conjugate distribution.

As an example, for binary data an appropriate data distribution is the Bernoulli distribution. The distribution is usually written as $p(x|\mu) = \mu^x(1-\mu)^{1-x}$, with $\mu$ in [0,1]. The exponential family form of this distribution, using the terms in equation (1) are: $h(x) = 0$, $\theta = \ln(\frac{\mu}{1-\mu})$ and $g(\theta) = -\ln(1 + e^\theta)$. The natural parameters can be mapped to the parameter values of the distribution using the link function, which is the logistic sigmoid in the case of the Bernoulli distribution. The terms of the conjugate distribution can also be derived easily.

## 3  Bayesian Exponential Family PCA

We can consider Bayesian Exponential Family PCA (BXPCA) as a method of searching for two matrices $\mathbf{V}$ and $\boldsymbol{\Theta}$, and we define the product matrix $\mathbf{P} = \mathbf{V}\boldsymbol{\Theta}$. In traditional PCA, the elements of the matrix $\mathbf{P}$ which are the means of Gaussians, lie in the same space as that of the data $\mathbf{X}$. In the case of BXPCA and other methods for non-Gaussian PCA such as EPCA [4], this matrix represents the *natural parameters* of the exponential family distribution of the data.

We represent the observed data as an $N \times D$ matrix $\mathbf{X} = \{\mathbf{x}_1, \dots, \mathbf{x}_N\}$, with an individual data point $\mathbf{x}_n = [x_{n1}, \dots, x_{nD}]$. $N$ is the number of data points and $D$ is the number of input features. $\boldsymbol{\Theta}$ is a $K \times D$ matrix with rows $\boldsymbol{\theta}_k$. $\mathbf{V}$ is a $N \times K$ matrix $\mathbf{V} = \{\mathbf{v}_1, \dots, \mathbf{v}_n\}$, and rows $\mathbf{v}_n = [v_{n1}, \dots, v_{nK}]$, are $K$-dimensional vectors of continuous values in $\mathbb{R}$. $K$ is the number of latent factors representing the dimensionality of the reduced space.

### 3.1  Model Specification

The generative process for the BXPCA model is described in figure 1. Let $\mathbf{m}$ and $\mathbf{S}$ be hyperparameters representing a $K$-dimensional vector of initial mean values and an initial covariance matrix respectively. Let $\alpha$ and $\beta$ be the hyperparameters corresponding to the shape and scale parameters of an inverse Gamma distribution. We start by drawing $\boldsymbol{\mu}$ from a Gaussian distribution and the elements $\sigma_k^2$ of the diagonal matrix $\boldsymbol{\Sigma}$ from an inverse gamma distribution:

$$\boldsymbol{\mu} \sim \mathcal{N}(\boldsymbol{\mu}|\mathbf{m}, \mathbf{S}) \qquad \sigma_k^2 \sim i\mathcal{G}(\alpha, \beta) \tag{3}$$

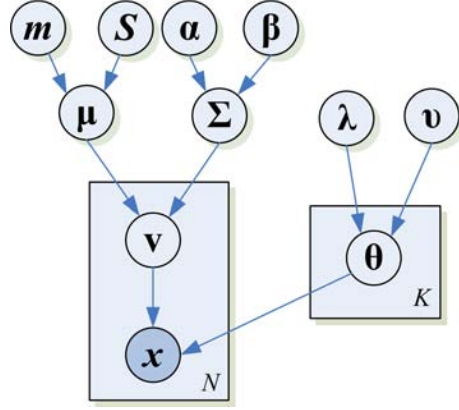

Figure 1: Graphical Model for Bayesian Exponential Family PCA.

For each data point $n$, we draw the $K$-dimensional entry $\mathbf{v}_n$ of the factor score matrix:

$$\mathbf{v}_n \sim \mathcal{N}(\mathbf{v}_n | \boldsymbol{\mu}, \boldsymbol{\Sigma}) \tag{4}$$

The data is described by an exponential family distribution with natural parameters $\boldsymbol{\theta}_k$. The exponential family distribution modelling the data, and the corresponding prior over the model parameters, is:

$$\mathbf{x}_n | \mathbf{v}_n, \boldsymbol{\Theta} \sim \text{Expon}\left(\sum_k v_{nk}\boldsymbol{\theta}_k\right) \qquad \boldsymbol{\theta}_k \sim \text{Conj}\left(\boldsymbol{\lambda}, \nu\right) \tag{5}$$

We denote $\boldsymbol{\Omega} = \{\mathbf{V}, \boldsymbol{\Theta}, \boldsymbol{\mu}, \boldsymbol{\Sigma}\}$ as the set of unknown parameters with hyperparameters $\boldsymbol{\Psi} = \{\mathbf{m}, \mathbf{S}, \alpha, \beta, \boldsymbol{\lambda}, \nu\}$. Given the graphical model, the joint probability of all parameters and variables is:

$$p(\mathbf{X}, \boldsymbol{\Omega} | \boldsymbol{\Psi}) = p(\mathbf{X} | \mathbf{V}, \boldsymbol{\Theta})p(\boldsymbol{\Theta} | \boldsymbol{\lambda}, \nu)p(\mathbf{V} | \boldsymbol{\mu}, \boldsymbol{\Sigma})p(\boldsymbol{\mu} | \mathbf{m}, \mathbf{S})p(\boldsymbol{\Sigma} | \alpha, \beta) \tag{6}$$

Using the model specification given by equations (3) - (5) and assuming that the parameter $\nu = 1$, the log-joint probability distribution is:

$$\ln p(\mathbf{X}, \boldsymbol{\Omega} | \boldsymbol{\Psi}) = \sum_{n=1}^{N}\left[\left(\sum_k v_{nk}\boldsymbol{\theta}_k\right)^{\top}\mathbf{x}_n + h(\mathbf{x}_n) + g\left(\sum_k v_{nk}\boldsymbol{\theta}_k\right)\right] \tag{7}$$

$$+ \sum_{k=1}^{K}\left[\boldsymbol{\lambda}^{\top}\boldsymbol{\theta}_k + g(\boldsymbol{\theta}_k) + f(\boldsymbol{\lambda})\right]$$

$$+ \sum_{n=1}^{N}\left[-\frac{K}{2}\ln(2\pi) - \frac{1}{2}\ln|\boldsymbol{\Sigma}| - \frac{1}{2}(\mathbf{v}_n - \boldsymbol{\mu})^T\boldsymbol{\Sigma}^{-1}(\mathbf{v}_n - \boldsymbol{\mu})\right]$$

$$- \frac{K}{2}\ln(2\pi) - \frac{1}{2}\ln|\boldsymbol{S}| - \frac{1}{2}(\boldsymbol{\mu} - \boldsymbol{m})^T\boldsymbol{S}^{-1}(\boldsymbol{\mu} - \boldsymbol{m})$$

$$+ \sum_{i=1}^{K}\left[\alpha\ln\beta - \ln\Gamma(\alpha) + (\alpha - 1)\ln\sigma_i^2 - \beta\sigma_i^2\right]$$

where the functions $h(\cdot)$, $g(\cdot)$ and $f(\cdot)$ correspond to the functions of the chosen conjugate distribution for the data.

## 3.2 Learning

The model parameters $\boldsymbol{\Omega} = \{\mathbf{V}, \boldsymbol{\Theta}, \boldsymbol{\mu}, \boldsymbol{\Sigma}\}$ are learned from the data using Hybrid Monte Carlo (HMC) sampling [7]. While the parameters $\boldsymbol{\Psi} = \{\mathbf{m}, \mathbf{S}, \alpha, \beta, \boldsymbol{\lambda}, \nu\}$ are treated as fixed hyperparameters, these can also be learned from the data. Hybrid Monte Carlo is a suitable sampler for use

with this model since all the variables are continuous and it is possible to compute the derivative of the log-joint probability. HMC is also an attractive scheme for sampling since it avoids the random walk behaviour of the Metropolis or the Gibbs sampling algorithms [7].

Hybrid Monte Carlo (HMC) is an auxiliary variable sampler where we sample from an augmented distribution $p(\mathbf{x}, \mathbf{u})$, rather than the target distribution $p(\mathbf{x})$, since it is easier to sample from this augmented distribution [8]. HMC utilises the gradient of the target distribution to improve mixing in high dimensions. In BXPCA, the target distribution is: $\mathcal{E}(\mathbf{\Omega}|\mathbf{\Psi}) = -\ln p(\mathbf{X}, \mathbf{\Omega}|\mathbf{\Psi})$ and represents the potential energy function. The auxiliary variable $\mathbf{u}$, is Gaussian and is used to define the kinetic energy $\mathcal{K} = \frac{1}{2}\mathbf{u}^T\mathbf{u}$. Furthermore, we define the gradient vector $\mathbf{\Delta}(\mathbf{X}, \mathbf{\Omega}) \triangleq \frac{\partial \mathcal{E}(\mathbf{\Omega})}{\partial \mathbf{\Omega}}$, which can be computed using equation (7). The sum of the kinetic and the potential energy defines the Hamiltonian. Samples of $\mathbf{\Omega}$ and $\mathbf{u}$ are obtained by combining the Hamiltonian with the gradient information in the simulation of so-called "leapfrog" steps. These details and the general pseudocode for HMC can be found in MacKay [9].

One key feature of HMC is that the dynamics is simulated in an unconstrained space. Therefore to correctly apply HMC to this model, we must ensure that all constrained variables are transformed to an unconstrained space, perform dynamics in this unconstrained space, and then transform the variables back to the original constrained space. The only variable that is constrained in BXPCA is $\mathbf{\Sigma}$ where each diagonal element $\sigma_k^2 > 0$. Each $\sigma_k^2$ can be transformed to a corresponding unconstrained variable $\xi_k$ using the transformation: $\sigma_k^2 = e^{\xi_k}$. This transformation requires that we then apply the chain rule for differentiation and that we must include the determinant of the Jacobian of the transformed variables, which is: $|\mathbf{J}| = \left|\frac{\partial}{\partial \xi_k}\exp(\sigma_k^2)\right| = |\exp(\xi_k)| = \sigma_k^2$.

We also extended the HMC procedure to handle missing inputs in a principled manner, by analytically integrating them out. In practice, this implies working with missing data under the Missing at Random (MAR) assumption. Here, we divide the data into the set of observed and missing data, $\mathbf{X} = \{\mathbf{X}^{obs}, \mathbf{X}^{missing}\}$, and use the set $\mathbf{X}^{obs}$ in the inference.

## 4   Related Work

**Exponential Family PCA:** Exponential family PCA (EPCA) [4] is a general class of PCA algorithms that allows the ideas of PCA to be applied to any data that can be modelled from a distribution in the exponential family. Like BXPCA, it is based on a factorisation of the data into a factor score matrix $\mathbf{V}$ and a factor loading matrix $\mathbf{\Theta}$. The algorithm is based on the optimisation of a loss function which is based on the Bregman divergence between the data and the learned reconstruction of the data. The learning is based on an alternating minimisation procedure where the two matrices $\mathbf{V}$ and $\mathbf{\Theta}$ are optimised in turn, and each optimisation is a convex function. The EPCA objective function can be seen as the likelihood function of a probabilistic model, and hence this optimisation corresponds to maximum a posteriori (MAP) learning. The use of MAP learning makes EPCA a deterministic latent variable model [6], since the latent variables are set to their MAP values.

In both our model and EPCA, the product $\mathbf{P} = \mathbf{V}\mathbf{\Theta}$ represents the natural parameters of the distribution over the data, and must be transformed using the link function to get to the parameter space of the associated data distribution. Our model is different from EPCA in that it is a fully probabilistic model in which all parameters can be integrated out by MCMC. Furthermore, EPCA does not include any form of regularisation and is prone to overfitting the data, which is avoided in the Bayesian framework. We will compare BXPCA to EPCA throughout this paper.

**Non-negative Matrix Factorisation:** Non-negative Matrix Factorisation (NMF) [2] is a technique of factorising a matrix into the product of two positive lower rank matrices. In NMF, the matrix product $\mathbf{P}$ approximates the *mean* parameters of the data distribution, and is thus in the same space as the data. A mean parameter for example, is the rate $\lambda$ if the data is modelled as a Poisson distribution, or is the probability of data being a 1 if the data is modelled as a Bernoulli. In NMF, $\mathbf{V}$ and $\mathbf{\Theta}$ are restricted to be positive matrices, and inference corresponds to maximum likelihood learning with a Poisson likelihood. Similarly to EPCA, this learning method places NMF in the class of deterministic latent variable methods.

**Discrete Components Analysis:** The Discrete Components Analysis (DCA) [3] is a family of probabilistic algorithms that deals with the application of PCA to discrete data and is a unification of the existing theory relating to dimensionality reduction with discrete distributions. In DCA the product $\mathbf{P} = \mathbf{V\Theta}$ is the *mean* parameter of the appropriate distribution over that data, as with NMF, and also constrains $\mathbf{V}$ and $\mathbf{\Theta}$ to be non-negative. The various algorithms of the DCA family are simulated using either Gibbs sampling or variational approximations.

**Bayesian Partial Membership:** The Bayesian Partial Membership (BPM) model is a clustering technique that allows data points to have fractional membership in multiple clusters. The model is derived from a finite mixture model which allows the usual indicator variables to take on any value in the range [0,1]. The resulting model has the same form as the model shown in figure 1, but instead of the model variable $\mathbf{V}$ being modelled as a Gaussian with unknown mean and covariance, it is instead modelled as a Dirichlet distribution. This difference is important, since it affects the interpretation of the results. In the BXPCA, we interpret the matrix $\mathbf{V}$ as a lower dimensional embedding of the data which can be used for dimensionality reduction. In contrast, the corresponding matrix for the BPM model, whose values are restricted to [0,1], is the partial membership of each data point and represents the extent to which each data point belongs to each of the $K$ clusters.

## 5   Results and Discussion

**Synthetic Data:** Synthetic data was generated by creating three 16-bit prototype vectors with each bit being generated with a probability of 0.5. Each of the three prototypes is replicated 200 times, resulting in a 600-point data set. We then flip bits in the replicates with a probability of 0.1, as in Tipping [10], thus adding noise about each of the prototypes. BXPCA inference was run using this data for 4000 iterations, using the first half as burn-in. Figure 2 demonstrates the learning process of BXPCA. In the initial phase of the sampling, the energy decreases slowly and the model is unable to learn any useful structure from the data. Around sample 750, the energy function decreases and some useful structure has been learnt. By sample 4000 the model has learnt the original data well, as can be seen by comparing sample 4000 and the original data.

To evaluate the performance of BXPCA, we define training and test data from the available

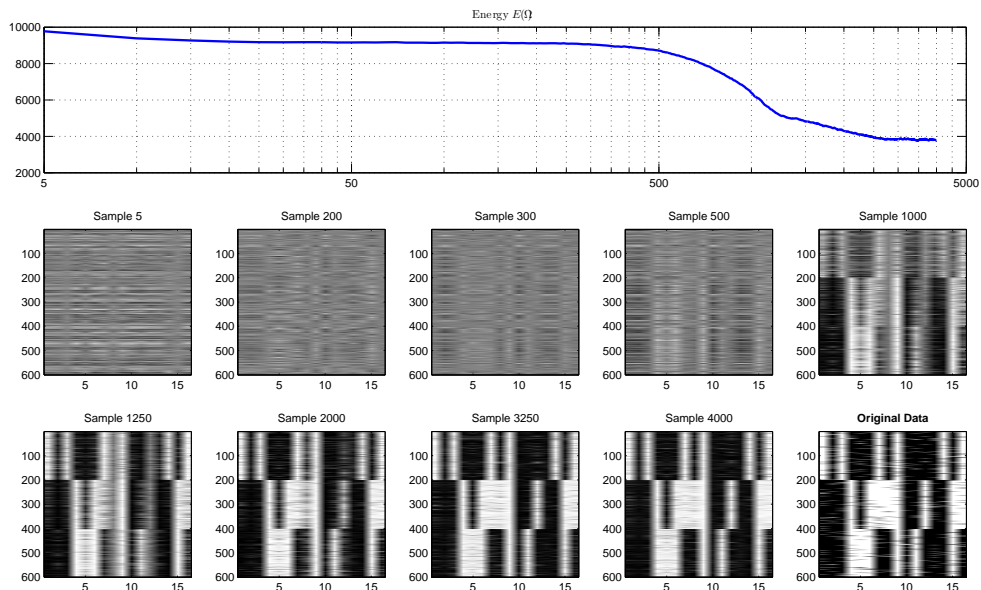

Figure 2: Reconstruction of data from samples at various stages of the sampling. The top plot shows the change in the energy function. The lower plots show the reconstructions and the original data.

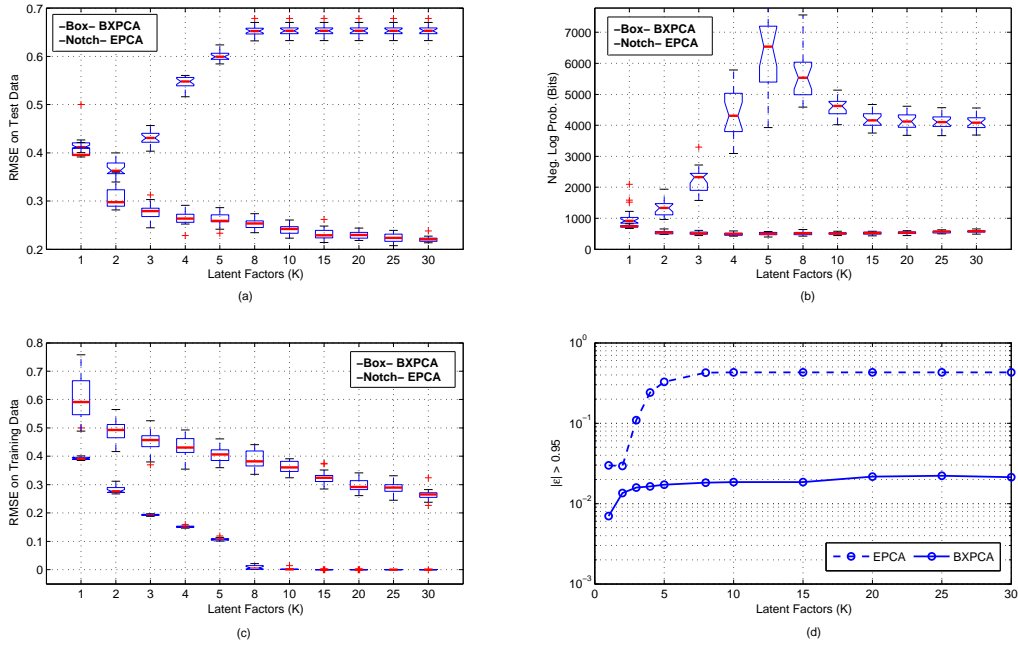

Figure 3: Boxplots comparing the NLP and RMSE of BXPCA and EPCA for various latent factors.

data. The test data was created by randomly selecting 10% of the data points. These test data points were set as missing values in the training data. Inference is then run using BXPCA, which has been extended to consider missing data. This method of using missing data is a natural way of testing these algorithms, since both are generative models. We calculate the negative log probability (NLP) and the root mean squared error (RMSE) using the testing data. We evaluate the same metrics for EPCA, which is also trained considering missing data. This missing data testing methodology is also used in the experiments on real data that are described later.

In figure 3a and 3b, the RMSE and NLP of the two algorithms are compared respectively, for various choices of the latent factor $K$. EPCA shows characteristic underfitting for $K = 1$ and demonstrates severe overfitting for large $K$. This overfitting is seen by the very large values of NLP for EPCA. If we examine the RMSE on the training data shown in figure 3c, we see the overfitting problem highlighted further, where the error on the training set is almost zero for EPCA, whereas BXPCA manages to avoid this problem. We expect that a random model would have a $NLP = 10\% \times 600 \times 16 = 960$ bits, but the NLP values for EPCA are significantly larger than this. This is because as EPCA begins to overfit, it becomes highly confident in its predictions and the proportion of bits which it believes are 1, for example, but which are actually 0, increases. This is shown in figure 3d, where we show the frequency of incorrect predictions, where the error between the predicted and actual bits is greater than 0.95. BXPCA, based on a Bayesian approach thus avoids overfitting and gives improved predictions.

**Digits Data:** BXPCA was applied to the CEDAR Buffalo digits dataset. The digit 2 was used, and consists of 700 greyscale images with 64 attributes. The digits were binarised by thresholding at a greyscale value of 128 from the 0 to 255 greyscale range. Table 1 compares the performance of BXPCA and EPCA, using the same method of creating training and testing data sets as for the synthetic data. BXPCA has lower RMSE and NLP than EPCA and also does not exhibit overfitting at large $K$, which can be seen in EPCA by the large value of NLP at $K = 5$.

**SPECT Data:** The data set describes the diagnosis of cardiac Single Proton Emission Computed Tomography (SPECT) images [11]. The data consists of 267 SPECT image sets, and has been processed resulting in 22 binary attributes. Table 2 compares the performance of BXPCA and EPCA. This dataset demonstrates that EPCA quickly overfits the data, as shown by the rapidly increasing values of NLP, and that the two algorithms perform equally well for low values of $K$.

Table 1: Table comparing BXPCA and EPCA on the digit 2 dataset.

| | K | 2 | 3 | 4 | 5 |
|---|---|---|---|---|---|
| BXPCA | NLP | 2032.3 | 2022.9 | 2002.4 | 2032.0 |
| | RMSE | **0.389** | **0.385** | **0.380** | **0.383** |
| EPCA | NLP | 2125.5 | 2482.1 | 2990.2 | 4708.8 |
| | RMSE | 0.392 | 0.393 | 0.399 | 0.402 |

Table 2: Table Comparing BXPCA and EPCA on the SPECT dataset.

| | K | 1 | 2 | 3 | 4 | 5 | 6 | 7 | 8 |
|---|---|---|---|---|---|---|---|---|---|
| BXPCA | NLP | 348.67 | 343.40 | 325.94 | 331.47 | 291.75 | 305.22 | 310.36 | 319.06 |
| | RMSE | 0.441 | 0.433 | **0.405** | **0.419** | **0.377** | **0.393** | **0.383** | **0.396** |
| EPCA | NLP | 388.18 | 516.78 | 507.79 | 1096.6 | 1727.4 | 4030.0 | 4209.0 | 4330.0 |
| | RMSE | **0.439** | **0.427** | 0.413 | 0.439 | 0.487 | 0.517 | 0.528 | 0.560 |

## 6  Choice of Final Embedding

For the purposes of dimensionality reduction, PCA is used to search for a low dimensional embedding $\mathbf{V}$ of the data points. In EPCA, the alternating minimisation returns a single $\mathbf{V}$ that is the low dimensional representation. In BXPCA though, we do not get a single $\mathbf{V}$, but rather many samples which represent the variation in the embedding. Furthermore, we cannot simply take the average of each of these samples to obtain a single $\mathbf{V}$, since we have not included any identifiability constraints in the model. This lack of identifiability subjects $\mathbf{V}$ to permutations of the columns, and to rotations of the matrix, making an average of the samples meaningless.

There are several approaches to obtaining a single low dimensional representation from the set of samples. The simplest approach is to choose from the set of available samples, the best global configuration, $\{\mathbf{V}^*, \mathbf{\Theta}^*\} = \arg\max_{\mathbf{\Omega}^{(s)}} p(\mathbf{X}, \mathbf{\Omega}^{(s)}|\mathbf{\Psi})$, and use this $\mathbf{V}^*$. A second approach aims to give further information about the variability of the embedding. We begin by fixing the model parameters to $\{\mathbf{\Theta}^*, \boldsymbol{\mu}^*, \mathbf{\Sigma}^*\}$. These can be set using the sample chosen in the first approach. We then sample $\mathbf{V}$ from the conditional distribution:

$$\mathbf{V} \sim p(\mathbf{V}|\mathbf{X}, \mathbf{\Theta}^*, \boldsymbol{\mu}^*, \mathbf{\Sigma}^*) \propto p(\mathbf{X}|\mathbf{V}, \mathbf{\Theta}^*)p(\mathbf{V}|\boldsymbol{\mu}^*, \mathbf{\Sigma}^*) \qquad (8)$$

where equation (8) is obtained using Bayes theorem and the joint probability distribution given in equation (6). We can now average these samples to obtain a single embedding since the problems of rotation and permutation have been removed by constraining the variables $\{\mathbf{\Theta}^*, \boldsymbol{\mu}^*, \mathbf{\Sigma}^*\}$. We demonstrate this procedure using the synthetic data described in the previous section for $K = 2$. Figure 4 shows the embedding in the 2D space for 10 data points and 20 independent samples drawn according to equation (8). The graph shows that there is some mean value and also gives us an understanding of the variation that is possible, in this 2D embedding. The drawback of this last approach is that it does not give any indication of the effect of variation in $\mathbf{\Theta}$. To gain some understanding of this effect, we can further extend this approach by choosing $Q$ random samples, $\mathbf{\Theta}^* = \{\mathbf{\Theta}^{*(1)}, \mathbf{\Theta}^{*(2)}, \ldots, \mathbf{\Theta}^{*(Q)}\}$, at convergence of the HMC sampler. We then repeat the aforementioned procedure for these various $\mathbf{\Theta}^{*(q)}$. This then gives an understanding of the variability of the final embedding, in terms of both $\mathbf{\Theta}$ and $\mathbf{V}$.

## 7  Conclusions and Future Work

We have described a Bayesian approach to PCA which is generalised to the exponential family. We have employed a Hybrid Monte Carlo sampling scheme with an energy based on the log-joint probability of the model. In particular, we have demonstrated the ability of BXPCA to learn the structure of the data while avoiding overfitting problems, which are experienced by other maximum likelihood approaches to exponential family PCA. We have demonstrated this using both synthetic and real data.

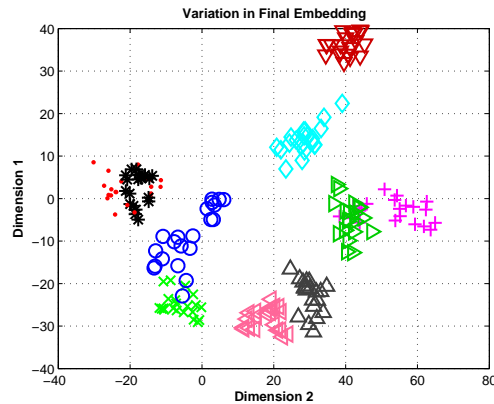

Figure 4: Variation in final embedding for 10 data points and various samples of $\mathbf{V}$

In future the model can be extended by considering an alternate distribution for the factor score matrix $\mathbf{V}$. Instead of considering a Gaussian distribution, a Laplacian or other heavy tailed distribution could be used, which would allow us to determine the lower dimensional embedding of the data, and also give the model a sparseness property. We could also specifically include restrictions on the form of the score and the loading matrices, $\mathbf{V}$ and $\mathbf{\Theta}$ respectively, to ensure identifiability. This makes learning in the model more complex since we must ensure that the restrictions are maintained. Also, it will prove interesting to consider alternate forms of inference, specifically the techniques of sequential Monte Carlo to allow for online inference.

**Acknowlegdements:** We thank Peter Gehler for the EPCA implementation. SM thanks the NRF SA and the Commonwealth Commission for support. KH was supported by an EPSRC Postdoctoral Fellowship (grant no. EP/E042694/1).

## References

[1] C. M. Bishop, *Pattern Recognition and Machine Learning*. Information Science and Statistics, Springer, August 2006.

[2] D. D. Lee and H. S. Seung, "Algorithms for non-negative matrix factorization," in *Advances in Neural Information Processing Systems*, vol. 13, pp. 556 – 562, MIT Press, Cambridge, MA, 2001.

[3] W. Buntine and A. Jakulin, "Discrete components analysis," in *Subspace, Latent Structure and Feature Selection*, vol. 3940/2006, pp. 1–33, Springer (LNCS), 2006.

[4] M. Collins, S. Dasgupta, and R. Schapire, "A generalization of principal components to the exponential family," in *Advances in Neural Information Processing Systems*, vol. 14, pp. 617 – 624, MIT Press, Cambridge, MA, 2002.

[5] Sajama and A. Orlitsky, "Semi-parametric exponential family PCA," in *Advances in Neural Information Processing Systems*, vol. 17, pp. 1177 – 1184, MIT Press, Cambridge, MA, 2004.

[6] M. Welling, C. Chemudugunta, and N. Sutter, "Deterministic latent variable models and their pitfalls," in *SIAM Conference on Data Mining (SDM)*, pp. 196 – 207, 2008.

[7] R. M. Neal, "Probabilistic inference using Markov Chain Monte Carlo methods," Tech. Rep. CRG-TR-93-1, University of Toronto, Department of Computer Science, 1993.

[8] C. Andrieu, N. De Freitas, A. Doucet, and M. I. Jordan, "An introduction to MCMC for machine learning," *Machine Learning*, vol. 50, pp. 5–43, 2003.

[9] D. J. C. MacKay, *Information Theory, Inference & Learning Algorithms*. Cambridge University Press, June 2002.

[10] M. E. Tipping, "Probabilistic visualisation of high dimensional binary data," in *Advances in Neural Information Processing Systems*, vol. 11, pp. 592 – 598, MIT Press, Cambridge, MA, 1999.

[11] "UCI machine learning repository." `http://archive.ics.uci.edu/ml/datasets/`.